# Combining causal and similarity-based reasoning

**Charles Kemp, Patrick Shafto, Allison Berke & Joshua B. Tenenbaum**
Department of Brain and Cognitive Sciences, MIT, Cambridge, MA 02139
{ckemp,shafto,berke,jbt}@mit.edu

## Abstract

Everyday inductive reasoning draws on many kinds of knowledge, including knowledge about relationships between properties and knowledge about relationships between objects. Previous accounts of inductive reasoning generally focus on just one kind of knowledge: models of causal reasoning often focus on relationships between properties, and models of similarity-based reasoning often focus on similarity relationships between objects. We present a Bayesian model of inductive reasoning that incorporates both kinds of knowledge, and show that it accounts well for human inferences about the properties of biological species.

## 1   Introduction

Will that berry taste good? Is that table strong enough to sit on? Predicting whether an object has an unobserved property is among the most basic of all inductive problems. Many kinds of knowledge appear to be relevant: different researchers emphasize the role of causal knowledge, similarity, category judgments, associations, analogical mappings, scripts, and intuitive theories, and each of these approaches accounts for an important subset of everyday inferences. Taken in isolation, however, each of these approaches is fundamentally limited. Humans draw on multiple kinds of knowledge and integrate them flexibly when required, and eventually our models should attempt to match this ability [1]. As an initial step towards this goal, we present a model of inductive reasoning that is sensitive both to causal relationships between properties and to similarity relationships between objects.

The inductive problem we consider can be formalized as the problem of filling in missing entries in an object-property matrix (Figure 1). Previous accounts of inductive reasoning generally address some version of this problem. Models of causal reasoning [2] usually focus on relationships between properties (Figure 1a): if animal $A$ has wings, for instance, it is likely that animal $A$ can fly. Similarity-based models [3, 4, 5] usually focus on relationships between objects (Figure 1b): if a duck carries gene $X$, a goose is probably more likely than a pig to carry the same gene. Previous models, however, cannot account for inferences that rely on similarity and causality: if a duck carries gene $X$ and gene $X$ causes enzyme $Y$ to be expressed, it is likely that a goose expresses enzyme $Y$ (Figure 1c). We develop a unifying model that handles inferences like this, and that subsumes previous probabilistic approaches to causal reasoning [2] and similarity-based reasoning [5, 6].

Our formal framework overcomes some serious limitations of the two approaches it subsumes. Approaches that rely on causal graphical models typically assume that the feature vectors of any two objects (any two rows of the matrix in Figure 1a) are conditionally independent given a causal network over the features. Suppose, for example, that the rows of the matrix correspond to people and the causal network states that smoking leads to lung cancer with probability 0.3. Suppose that Tim, Tom and Zach are smokers, that Tim and Tom are identical twins, and that Tim has lung cancer. The assumption of conditional independence implies that Tom and Zach are equally likely to suffer from long cancer, a conclusion that seems unsatisfactory. The assumption is false because of variables that are unknown but causally relevant—variables capturing unknown biological and environmental factors that mediate the relationship between smoking and disease. Dealing with these unknown

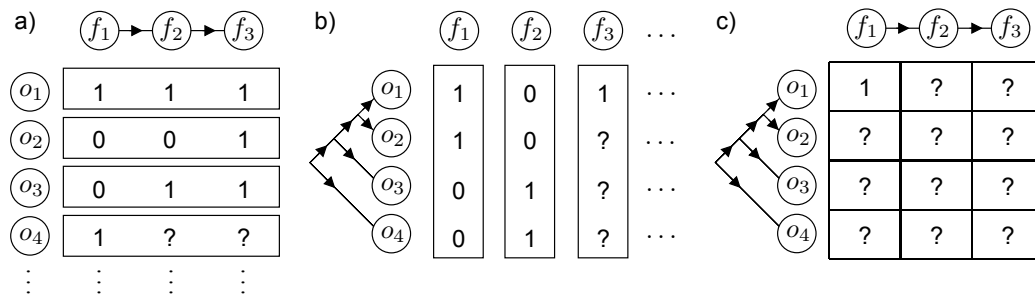

Figure 1: (a) Models of causal reasoning generally assume that the rows of an object-feature matrix are conditionally independent given a causal structure over the features. These models are often used to make predictions about unobserved features of novel objects. (b) Models of similarity-based reasoning generally assume that columns of the matrix are conditionally independent given a similarity structure over the objects. These models are often used to make predictions about novel features. (c) We develop a generative model for object-feature matrices that incorporates causal relationships between features and similarity relationships between objects. The model uses both kinds of information to make predictions about matrices with missing entries.

variables is difficult, but we suggest that knowledge about similarity between objects can help. Since Tim is more similar to Tom than Zach, our model correctly predicts that Tom is more likely to have lung cancer than Zach.

Previous models of similarity-based reasoning [5, 6] also suffer from a restrictive assumption of conditional independence. This time the assumption states that features (columns of the matrix in Figure 1b) are conditionally independent given information about the similarity between objects. Empirical tests of similarity-based models often attempt to satisfy this assumption by using *blank properties*—subjects, for example, might be told that coyotes have property P, and asked to judge the probability that foxes have property P [3]. To a first approximation, inferences in tasks like this conform to judgments of similarity: subjects conclude, for example, that foxes are more likely to have property P than mice, since coyotes are more similar to foxes than mice. People, however, find it natural to reason about properties that are linked to familiar properties, and that therefore violate the assumption of conditional independence. Suppose, for instance, you learn that desert foxes have skin that is resistant to sunburn. It now seems that desert rats are more likely to share this property than arctic foxes, even though desert foxes are more similar in general to arctic foxes than to desert rats. Our model captures inferences like this by incorporating causal relationships between properties: in this case, having sunburn-resistant skin is linked to the property of living in the desert.

Limiting assumptions of conditional independence can be avoided by specifying a joint distribution on an entire object-property matrix. Our model uses a distribution that is sensitive both to causal relationships between properties and to similarity relationships between objects. We know of no previous models that attempt to combine causality and similarity, and one set of experiments that has been taken to suggest that people find it difficult to combine these sources of information [7]. After introducing our model, we present two experiments designed to test it. The results suggest that people are able to combine causality with similarity, and that our model accounts well for this capacity.

## 2 A generative model for object-feature matrices

Consider first a probabilistic approach to similarity-based reasoning. Assume that $S_o$ is an object structure: a graphical model that captures relationships between a known set of objects (Figure 1b). Suppose, for instance, that the objects include a mouse, a rat, a squirrel and a sheep ($o_1$ through $o_4$). $S_o$ can be viewed as a graphical model that captures phylogenetic relationships, or as a formalization of the intuitive similarity between these animals. Given some feature of interest, the feature values for all objects can be collected into an object vector $v_o$ and $S_o$ specifies a distribution $P(v_o)$ on these vectors.

We work with the case where $(S_o, \lambda)$ is a tree-structured graphical model of the sort previously used by methods for Bayesian phylogenetics [8] and cognitive models of property induction [5, 6]. The objects lie at the leaves of the tree, and we assume that object vectors are binary vectors generated by a mutation process over the tree. This process has a parameter, $\lambda$, that represents the base rate of a novel feature—the expected proportion of objects with that feature. For instance, if $\lambda$ is low, the model $(S_o, \lambda)$ will predict that a novel feature will probably not be found in any of the animals, but if the feature does occur in exactly two of the animals, the mouse and the rat are a more likely pair than the mouse and the sheep.

The mutation process can be formalized as a continuous-time Markov process with two states (off and on) and with infinitesimal matrix:

$$Q = \begin{bmatrix} -\lambda & \lambda \\ 1 - \lambda & -(1 - \lambda) \end{bmatrix}$$

We can generate object vectors from this model by imagining a binary feature spreading out over the tree from root to leaves. The feature is on at the root with probability $\lambda$, and the feature may switch states at any point along any branch. The parameter $\lambda$ determines how easy it is to move between the on state and the off state. If $\lambda$ is high, it will be easy for the Markov process to enter the on state, and difficult for it to leave once it is there.

Consider now a probabilistic approach to causal reasoning. Assume that $S_f$ is a feature structure: a graphical model that captures relationships between a known set of features (Figure 1a). The features, for instance, may correspond to enzymes, and $S_f$ may capture the causal relationships between these enzymes. One possible structure states that enzyme $f_1$ is involved in the production of enzyme $f_2$, which is in turn involved in the production of enzyme $f_3$. The feature values for any given object can be collected into a feature vector $v_f$ and $S_f$ specifies a distribution $P(v_f)$ on these vectors.

Suppose now that we are interested in a model that combines the knowledge represented by $S_f$ and $S_o$ (Figure 1c). Given that the mouse expresses enzyme $f_1$, for instance, a combined model should predict that rats are more likely than squirrels to express enzyme $f_2$. Formally, we seek a distribution $P(M)$, where $M$ is an object-feature matrix, and $P(M)$ is sensitive to both the relationships between features and the relationships between animals. Given this distribution, Bayesian inference can be used to make predictions about the missing entries in a partially observed matrix.

If the features in $S_f$ happen to be independent (Figure 1b), we can assume that column $i$ of the matrix is generated by $(S_o, \lambda_i)$, where $\lambda_i$ is the base rate of $f_i$. Consider then the case where $S_f$ captures causal relationships between the features (Figure 1c). These causal relationships will typically depend on several hidden variables. Causal relationships between enzymes, for instance, are likely to depend on other biological variables, and the causal link between smoking and lung cancer is mediated by many genetic and environmental variables. Often little is known about these hidden variables, but to a first approximation we can assume that they respect the similarity structure $S_o$. In Figure 1c, for example, the unknown variables that mediate the relationship between $f_1$ and $f_2$ are more likely to take the same value in $o_1$ and $o_2$ than in $o_1$ and $o_4$.

We formalize these intuitions by converting a probabilistic model $S_f$ (Figure 2a) into an equivalent model $S_f^D$ (Figure 2b) that uses a deterministic combination of independent random events. These random events will include hidden but causally relevant variables. In Figure 2b, for example, the model $S_f^D$ indicates that the effect $e$ is deterministically present if the cause $c$ is present and the transmission mechanism $t$ is active, or if there is a background cause $b$ that activates $e$. The model $S_f^D$ is equivalent to $S_f$ in the sense that both models induce the same distribution over the variables that appear in $S_f$. In general there will be many models $S_f^D$ that meet this condition, and there are algorithms which convert $S_f$ into one of these models [9]. For some applications it might be desirable to integrate over all of these models, but here we attempt to choose the simplest—the model $S_f^D$ with the fewest variables.

Given a commitment to a specific deterministic model, we assume that the root variables in $S_f^D$ are independently generated over $S_o$. More precisely, suppose that the base rate of the $i$th variable in $S_f^D$ is $\lambda_i$. The distribution $P(M)$ we seek must meet two conditions (note that each candidate matrix $M$ now has a column for each variable in $S_f^D$). First, the marginal distribution on each row must match

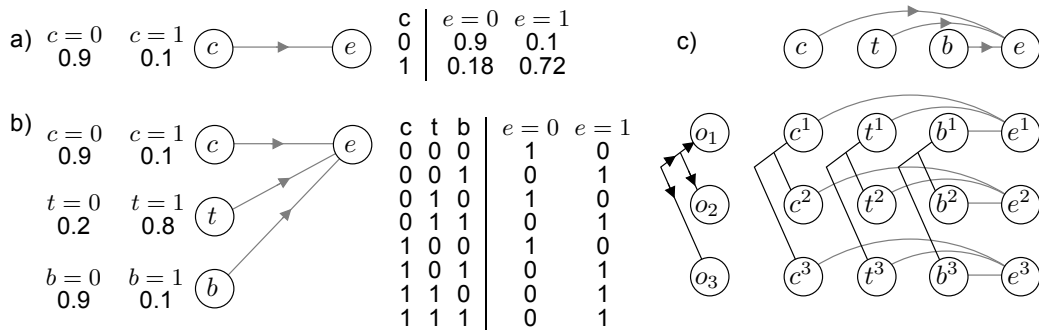

Figure 2: (a) A graphical model $S_f$ that captures a probabilistic relationship between a cause $c$ and an effect $e$. (b) A deterministic model $S_f^D$ that induces the same joint distribution over $c$ and $e$. $t$ indicates whether the mechanism of causal transmission between $c$ and $e$ is active, and $b$ indicates whether $e$ is true owing to a background cause independent of $c$. All of the root variables ($c$, $t$ and $b$) are independent, and the remaining variables ($e$) are deterministically specified once the root variables are fixed. (c) A graphical model created by combining $S_f^D$ with a tree-structured representation of the similarity between three objects. The root variables in $S_f^D$ ($c$, $t$, and $b$) are independently generated over the tree. Note that the arrows on the edges of the combined model have been suppressed.

the distribution specified by $S_f^D$. Second, if $f_i$ is a root variable in $S_f^D$, the marginal distribution on column $i$ must match the distribution specified by $(S_o, \lambda_i)$.

There is precisely one distribution $P(M)$ that satisfies these conditions, and we can represent it using a graphical model that we call the combined model. Suppose that there are $n$ objects in $S_o$. To create the combined model, we first introduce $n$ copies of $S_f^D$. For each root variable $i$ in $S_f^D$, we now connect all copies of variable $i$ according to the structure of $S_o$ (Figure 2c). The resulting graph provides the topology of the combined model, and the conditional probability distributions (CPDs) are inherited from $S_o$ and $S_f^D$. Each node that belongs to the $i$th copy of $S_o$ inherits a CPD from $(S_o, \lambda_i)$, and all remaining nodes inherit a (deterministic) CPD from $S_f$. Now that the distribution $P(M)$ is represented as a graphical model, standard inference techniques can be used to compute the missing entries in a partially-observed matrix $M$. All results in this paper were computed using the implementation of the junction tree algorithm included in the Bayes Net toolbox [10].

## 3 Experiments

When making inductive inferences, a rational agent should exploit all of the information available, including causal relationships between features and similarity relationships between objects. Whether humans are able to meet this normative standard is not clear, and almost certainly varies from task to task. On one hand, there are motivating examples like the case of the three smokers where it seems natural to think about causal relationships and similarity relationships at the same time. On the other hand, Rehder [7] argues that causal information tends to overwhelm similarity information, and supports this conclusion with data from several tasks involving artificial categories. To help resolve these competing views, we designed several tasks where subjects were required to simultaneously reason about causal relationships between enzymes and similarity relationships between animals.

### 3.1 Experiment 1

**Materials and Methods.** 16 adults participated in this experiment. Subjects were asked to reason about the presence of enzymes in a set of four animals: a mouse, a rat, a sheep, and a squirrel. Each subject was trained on two causal structures, each of which involved three enzymes. Pseudo-biological names like "dexotase" were used in the experiment, but here we will call the enzymes $f_1$, $f_2$ and $f_3$. In the chain condition, subjects were told that $f_3$ is known to be produced by several

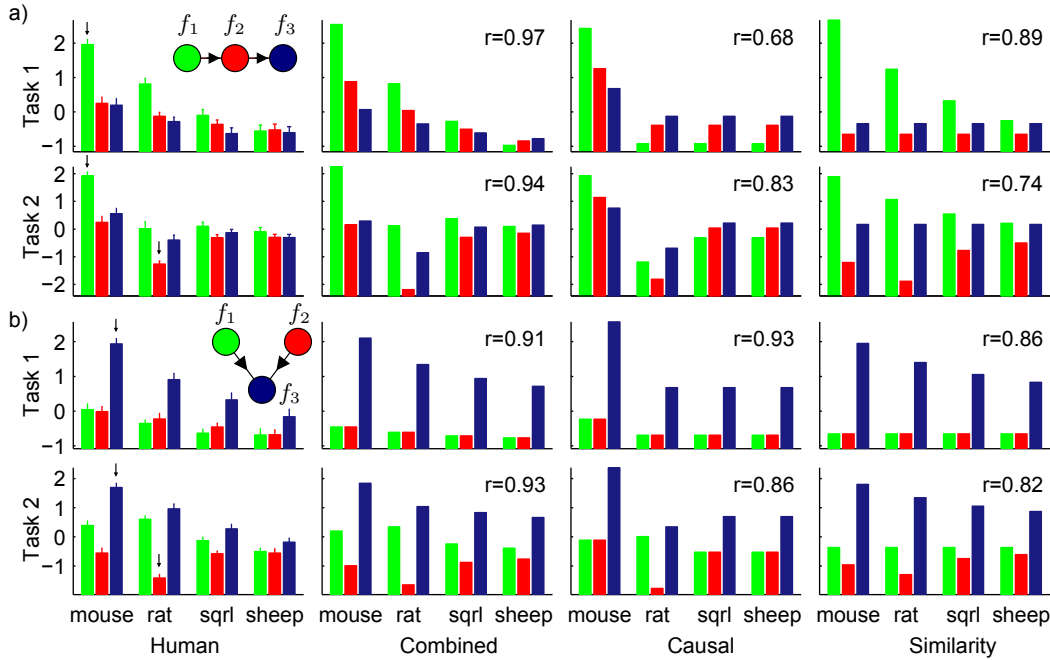

Figure 3: Experiment 1: Behavioral data (column 1) and predictions for three models. (a) Results for the chain condition. Known test results are marked with arrows: in task 1, subjects were told only that the mouse had tested positive for $f_1$, and in task 2 they were told in addition that the rat had tested negative for $f_2$. Error bars represent the standard error of the mean. (b) Results for the common-effect condition.

pathways, and that the most common pathway begins with $f_1$, which stimulates production of $f_2$, which in turn leads to the production of $f_3$. In the common-effect condition, subjects were told that $f_3$ is known to be produced by several pathways, and that one of the most common pathways involves $f_1$ and the other involves $f_2$.

To reinforce each causal structure, subjects were shown 20 cards representing animals from twenty different mammal species (names of the species were not supplied). The card for each animal represented whether that animal had tested positive for each of the three enzymes. The cards were chosen to be representative of the distribution captured by a causal network with known structure (chain or common-effect) and known parameterization. In the chain condition, for example, the network was a noisy-or network with the form of a chain, where leak probabilities were set to 0.4 ($f_1$) or 0.3 ($f_2$ and $f_3$), and the probability that each causal link was active was set to 0.7. After subjects had studied the cards for as long as they liked, the cards were removed and subjects were asked several questions about the enzymes (e.g. "you learn about a new mammal—how likely is it that the mammal produces $f_3$?") The questions in this training phase were intended to encourage subjects to reflect on the causal relationships between the enzymes.

In both conditions, subjects were told that they would be testing the four animals (mouse, rat, sheep and squirrel) for each of the three enzymes. Each condition included two tasks. In the chain condition, subjects were told that the mouse had tested positive for $f_1$, and asked to predict the outcome of each remaining test (Figure 1c). Subjects were then told in addition that the rat had tested negative for $f_2$, and again asked to predict the outcome of each remaining test. Note that this second task requires subjects to integrate causal reasoning with similarity-based reasoning: causal reasoning predicts that the mouse has $f_2$, and similarity-based reasoning predicts that it does not. In the common-effect condition, subjects were told that the mouse had tested positive for $f_3$, then told in addition that the rat had tested negative for $f_2$. Ratings were provided on a scale from 0 (very likely to test negative) to 100 (very likely to test positive).

**Results.** Subjects used the 100 point scale very differently: in task 1 of each condition, some subjects chose numbers between 80 and 100, and others chose numbers between 0 and 100. We

therefore converted each set of ratings to z-scores. Average z-scores are shown in the first column of Figure 3, and the remaining columns show predictions for several models. In each case, model predictions have been converted from probabilities to z-scores to allow a direct comparison with the human data.

Our combined model uses a tree over the four animals and a causal network over the features. We used the tree shown in Figure 1b, where objects $o_1$ through $o_4$ correspond to the mouse, the rat, the squirrel and the sheep. The tree component of our model has one free-parameter—the total path length of the tree. The smaller the path length, the more likely that all four animals have the same feature values, and the greater the path length, the more likely that distant animals in the tree (e.g. the mouse and the sheep) will have different feature values. All results reported here use the same value of this parameter—the value that maximizes the average correlation achieved by our model across Experiments 1 and 2. The causal component of our model includes no free parameters, since we used the parameters of the network that generated the cards shown to subjects during the training phase. Comparing the first two columns of Figure 3, we see that our combined model accounts well for the human data.

Columns 3 and 4 of Figure 3 show model predictions when we remove the similarity component (column 3) or the causal component (column 4) from our combined model. The model that uses the causal network alone is described by [2], among others, and the model that uses the tree alone is described by [6]. Both of these models miss qualitative trends evident in the human data. In task 1 of each condition, the causal model makes identical predictions about the rat, the squirrel and the sheep: in task 1 of the chain condition, for example, it cannot use the similarity between the mouse and the rat to predict that the rat is also likely to test positive for $f_1$. In task 1 of each condition the similarity model predicts that the unobserved features ($f_2$ and $f_3$ for the chain condition, and $f_1$ and $f_2$ for the common-effect condition) are distributed identically across the four animals. In task 1 of the chain condition, for example, the similarity model does not predict that the mouse is more likely than the sheep to test positive for $f_2$ and $f_3$.

The limitations of the causal and similarity models suggest that some combination of causality and similarity is necessary to account for our data. There are likely to be approaches other than our combined model that account well for our data, but we suggest that accurate predictions will only be achieved when the causal network and the similarity information are tightly integrated. Simply averaging the predictions for the causal model and the similarity model will not suffice: in task 1 of the chain condition, for example, both of these models predict that the rat and the sheep are equally likely to test positive for $f_2$, and computing an average across these models will result in the same prediction.

## 3.2   Experiment 2

Our working hypothesis is that similarity and causality should be combined in most contexts. An alternative hypothesis—the *root-variables* hypothesis—was suggested to us by Bob Rehder, and states that similarity relationships are used only if some of the root variables in a causal structure $S_f$ are unobserved. For instance, similarity might have influenced inferences in the chain condition of Experiment 1 only because the root variable $f_1$ was never observed for all four animals.

The root-variables hypothesis should be correct in cases where all root variables in the *true* causal structure are known. In Figure 2c, for instance, similarity no longer plays a role once the root variables are observed, since the remaining variables are deterministically specified. We are interested, however, in cases where $S_f$ may not contain all of the causally relevant variables, and where similarity can help to make predictions about the effects of unobserved variables. Consider, for example, the case of the three smokers, where $S_f$ states that smoking causes lung cancer. Even though the root variable is observed for Tim, Tom and Zach (all three are smokers), we still believe that Tom is more likely to suffer from lung cancer than Zach having discovered that Tim has lung cancer. The case of the three smokers therefore provides intuitive evidence against the root-variables hypothesis, and we designed a related experiment to explore this hypothesis empirically.

**Materials and Methods.** Experiment 2 was similar to Experiment 1, except that the common-effect condition was replaced by a common-cause condition. In the first task for each condition, subjects were told only that the mouse had tested positive for $f_1$. In the second task, subjects were told in addition that the rat, the squirrel and the sheep had tested positive for $f_1$, and that the mouse had

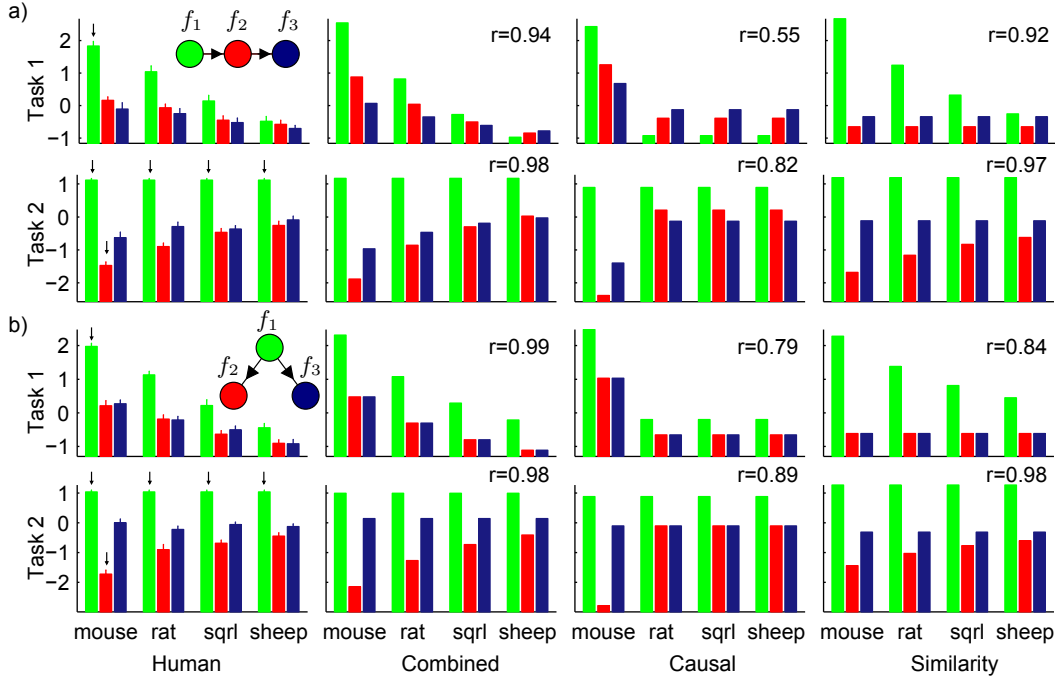

Figure 4: Experiment 2: Behavioral data and predictions for three models. In task 2 of each condition, the root variable in the causal network ($f_1$) is observed for all four animals.

tested negative for $f_2$. Note that in the second task, values for the root variable ($f_1$) were provided for all animals. 18 adults participated in this experiment.

**Results.** Figure 4 shows mean z-scores for the subjects and for the four models described previously. The judgments for the first task in each condition replicate the finding from Experiment 1 that subjects combine causality and similarity when just one of the 12 animal-feature pairs is observed. The results for the second task rule out the root-variables hypothesis. In the chain condition, for example, the causal model predicts that the rat and the sheep are equally likely to test positive for $f_2$. Subjects predict that the rat is less likely than the sheep to test positive for $f_2$, and our combined model accounts for this prediction.

## 4 Discussion

We developed a model of inductive reasoning that is sensitive to causal relationships between features and similarity relationships between objects, and demonstrated in two experiments that it provides a good account of human reasoning. Our model makes three contributions. First, it provides an integrated view of two inductive problems—causal reasoning and similarity-based reasoning—that are usually considered separately. Second, unlike previous accounts of causal reasoning, it acknowledges the importance of unknown but causally relevant variables, and uses similarity to constrain inferences about the effects of these variables. Third, unlike previous models of similarity-based reasoning, our model can handle novel properties that are causally linked to known properties.

For expository convenience we have emphasized the distinction between causality and similarity, but the notion of similarity needed by our approach will often have a causal interpretation. A tree-structured taxonomy, for example, is a simple representation of the causal process that generated biological species—the process of evolution. Our combined model can therefore be seen as a causal model that takes both relationships between features and evolutionary relationships between species into account. More generally, our framework can be seen as a method for building sophisticated causal models, and our experiments suggest that these kinds of models will be needed to account for the complexity and subtlety of human causal reasoning.

Other researchers have proposed strategies for combining probabilistic models [11], and some of these methods may account well for our data. In particular, the product of experts approach [12] should lead to predictions that are qualitatively similar to the predictions of our combined model. Unlike our approach, a product of experts model is not a directed graphical model, and does not support predictions about interventions. Neither of our experiments explored inferences about interventions, but an adequate causal model should be able to handle inferences of this sort.

Causal knowledge and similarity are just two of the many varieties of knowledge that support inductive reasoning. Any single form of knowledge is a worthy topic of study, but everyday inferences often draw upon multiple kinds of knowledge. We have not provided a recipe for combining arbitrary forms of knowledge, but our work illustrates two general themes that may apply quite broadly. First, different generative models may capture different aspects of human knowledge, but all of these models use a common language: the language of probability. Probabilistic models are modular, and can be composed in many different ways to build integrated models of inductive reasoning. Second, the stochastic component of most generative models is in part an expression of ignorance. Using one model (e.g. a similarity model) to constrain the stochastic component of another model (e.g. a causal network) may be a relatively general method for combining probabilistic knowledge representations.

Although we have focused on human reasoning, integrated models of induction are needed in many scientific fields. Our combined model may find applications in computational biology: predicting whether an organism expresses a certain gene, for example, should rely on phylogenetic relationships between organisms and causal relationships between genes. Related models have already been explored: Engelhardt et al. [13] develop an approach to protein function prediction that combines phylogenetic relationships between proteins with relationships between protein functions, and several authors have explored models that combine phylogenies with hidden Markov models. Combining two models is only a small step towards a fully integrated approach, but probability theory provides a *lingua franca* for combining many different representations of the world.

**Acknowledgments** We thank Bob Rehder and Brian Milch for valuable discussions. This work was supported in part by AFOSR MURI contract FA9550-05-1-0321, the William Asbjornsen Albert memorial fellowship (CK) and the Paul E. Newton Chair (JBT).

# References

[1] A. Newell. *Unified theories of cognition*. Harvard University Press, Cambridge, MA, 1989.

[2] B. Rehder and R. Burnett. Feature inference and the causal structure of categories. *Cognitive Science*, 50: 264–314, 2005.

[3] D. N. Osherson, E. E. Smith, O. Wilkie, A. Lopez, and E. Shafir. Category-based induction. *Psychological Review*, 97(2):185–200, 1990.

[4] S. A. Sloman. Feature-based induction. *Cognitive Psychology*, 25:231–280, 1993.

[5] C. Kemp and J. B. Tenenbaum. Theory-based induction. In *Proceedings of the Twenty-Fifth Annual Conference of the Cognitive Science Society*, pages 658–663. Lawrence Erlbaum Associates, 2003.

[6] C. Kemp, T. L. Griffiths, S. Stromsten, and J. B. Tenenbaum. Semi-supervised learning with trees. In *Advances in Neural Information Processing Systems 16*. MIT Press, Cambridge, MA, 2004.

[7] B. Rehder. When similarity and causality compete in category-based property generalization. *Memory and Cognition*, 34(1):3–16, 2006.

[8] J. P. Huelsenbeck and F. Ronquist. MRBAYES: Bayesian inference of phylogenetic trees. *Bioinformatics*, 17(8):754–755, 2001.

[9] D. Poole. Probabilistic Horn abduction and Bayesian networks. *Artificial Intelligence*, 64(1):81–129, 1993.

[10] K. Murphy. The Bayes Net Toolbox for MATLAB. *Computing science and statistics*, 33:1786–1789, 2001.

[11] C. Genest and J. V. Zidek. Combining probability distributions: a critique and an annotated bibliography. *Statistical Science*, 1(2):114–135, 1986.

[12] G. E. Hinton. Modelling high-dimensional data by combining simple experts. In *Proceedings of the 17th National Conference on Artificial Intelligence*. AAAI Press, 2000.

[13] B. E. Engelhardt, M. I. Jordan, and S. E. Brenner. A graphical model for predicting protein molecular function. In *Proceedings of the 23rd International Conference on Machine Learning*, 2006.
